# Multi-task Gaussian Process Prediction

**Edwin V. Bonilla,   Kian Ming A. Chai,   Christopher K. I. Williams**
School of Informatics, University of Edinburgh, 5 Forrest Hill, Edinburgh EH1 2QL, UK
edwin.bonilla@ed.ac.uk, K.M.A.Chai@sms.ed.ac.uk, c.k.i.williams@ed.ac.uk

## Abstract

In this paper we investigate multi-task learning in the context of Gaussian Processes (GP). We propose a model that learns a shared covariance function on input-dependent features and a "free-form" covariance matrix over tasks. This allows for good flexibility when modelling inter-task dependencies while avoiding the need for large amounts of data for training. We show that under the assumption of noise-free observations and a block design, predictions for a given task only depend on its target values and therefore a cancellation of inter-task transfer occurs. We evaluate the benefits of our model on two practical applications: a compiler performance prediction problem and an exam score prediction task. Additionally, we make use of GP approximations and properties of our model in order to provide scalability to large data sets.

## 1   Introduction

Multi-task learning is an area of active research in machine learning and has received a lot of attention over the past few years. A common set up is that there are multiple related tasks for which we want to avoid tabula rasa learning by sharing information across the different tasks. The hope is that by learning these tasks simultaneously one can improve performance over the "no transfer" case (i.e. when each task is learnt in isolation). However, as pointed out in [1] and supported empirically by [2], assuming relatedness in a set of tasks and simply learning them together can be detrimental. It is therefore important to have models that will generally benefit related tasks and will not hurt performance when these tasks are unrelated. We investigate this in the context of Gaussian Process (GP) prediction.

We propose a model that attempts to learn inter-task dependencies based solely on the *task identities* and the observed data for each task. This contrasts with approaches in [3, 4] where task-descriptor features $\mathbf{t}$ were used in a parametric covariance function over different tasks—such a function may be too constrained by both its parametric form and the task descriptors to model task similarities effectively. In addition, for many real-life scenarios task-descriptor features are either unavailable or difficult to define correctly. Hence we propose a model that learns a "free-form" task-similarity matrix, which is used in conjunction with a parameterized covariance function over the input features $\mathbf{x}$.

For scenarios where the number of input observations is small, multi-task learning augments the data set with a number of different tasks, so that model parameters can be estimated more confidently; this helps to minimize over-fitting. In our model, this is achieved by having a common covariance function over the features $\mathbf{x}$ of the input observations. This contrasts with the semiparametric latent factor model [5] where, with the same set of input observations, one has to estimate the parameters of several covariance functions belonging to different latent processes.

For our model we can show the interesting theoretical property that there is a cancellation of inter-task transfer in the specific case of noise-free observations and a block design. We have investigated both gradient-based and EM-based optimization of the marginal likelihood for learning the hyper-parameters of the GP. Finally, we make use of GP approximations and properties of our model in

order to scale our approach to large multi-task data sets, and evaluate the benefits of our model on two practical multi-task applications: a compiler performance prediction problem and a exam score prediction task.

The structure of the paper is as follows: in section 2 we outline our model for multi-task learning, and discuss some approximations to speed up computations in section 3. Related work is described in section 4. We describe our experimental setup in section 5 and give results in section 6.

## 2 The Model

Given a set $X$ of $N$ distinct inputs $\mathbf{x}_1, \ldots, \mathbf{x}_N$ we define the complete set of responses for $M$ tasks as $\mathbf{y} = (y_{11}, \ldots, y_{N1}, \ldots, y_{12}, \ldots, y_{N2}, \ldots, y_{1M}, \ldots, y_{NM})^{\mathrm{T}}$, where $y_{il}$ is the response for the $l^{\text{th}}$ task on the $i^{\text{th}}$ input $\mathbf{x}_i$. Let us also denote the $N \times M$ matrix $Y$ such that $\mathbf{y} = \operatorname{vec} Y$.

Given a set of observations $\mathbf{y}_o$, which is a subset of $\mathbf{y}$, we want to predict some of the unobserved response-values $\mathbf{y}_u$ at some input locations for certain tasks.

We approach this problem by placing a GP prior over the latent functions $\{f_l\}$ so that we directly induce correlations between tasks. Assuming that the GPs have zero mean we set

$$\langle f_l(\mathbf{x}) f_k(\mathbf{x}') \rangle = K_{lk}^f k^x(\mathbf{x}, \mathbf{x}') \qquad y_{il} \sim \mathcal{N}(f_l(\mathbf{x}_i), \sigma_l^2), \tag{1}$$

where $K^f$ is a positive semi-definite (PSD) matrix that specifies the inter-task similarities, $k^x$ is a covariance function over inputs, and $\sigma_l^2$ is the noise variance for the $l^{\text{th}}$ task. Below we focus on *stationary* covariance functions $k^x$; hence, to avoid redundancy in the parametrization, we further let $k^x$ be only a *correlation* function (i.e. it is constrained to have unit variance), since the variance can be explained fully by $K^f$.

The important property of this model is that the joint Gaussian distribution over $\mathbf{y}$ is not block-diagonal wrt tasks, so that observations of one task can affect the predictions on another task. In [4, 3] this property also holds, but instead of specifying a general PSD matrix $K^f$, these authors set $K_{lk}^f = k^f(\mathbf{t}_l, \mathbf{t}_k)$, where $k^f(\cdot, \cdot)$ is a covariance function over the task-descriptor features $\mathbf{t}$.

One popular setup for multi-task learning is to assume that tasks can be clustered, and that there are inter-task correlations between tasks in the same cluster. This can be easily modelled with a general task-similarity $K^f$ matrix: if we assume that the tasks are ordered with respect to the clusters, then $K^f$ will have a block diagonal structure. Of course, as we are learning a "free form" $K^f$ the ordering of the tasks is irrelevant in practice (and is only useful for explanatory purposes).

### 2.1 Inference

Inference in our model can be done by using the standard GP formulae for the mean and variance of the predictive distribution with the covariance function given in equation (1). For example, the mean prediction on a new data-point $\mathbf{x}_*$ for task $l$ is given by

$$\bar{f}_l(\mathbf{x}_*) = (\mathbf{k}_l^f \otimes \mathbf{k}_*^x)^T \Sigma^{-1} \mathbf{y} \qquad\qquad \Sigma = K^f \otimes K^x + D \otimes I \tag{2}$$

where $\otimes$ denotes the Kronecker product, $\mathbf{k}_l^f$ selects the $l^{\text{th}}$ column of $K^f$, $\mathbf{k}_*^x$ is the vector of covariances between the test point $\mathbf{x}_*$ and the training points, $K^x$ is the matrix of covariances between all pairs of training points, $D$ is an $M \times M$ diagonal matrix in which the $(l, l)^{\text{th}}$ element is $\sigma_l^2$, and $\Sigma$ is an $MN \times MN$ matrix.

In section 2.3 we show that when there is no noise in the data (i.e. $D = \mathbf{0}$), there will be no transfer between tasks.

### 2.2 Learning Hyperparameters

Given the set of observations $\mathbf{y}_o$, we wish to learn the parameters $\boldsymbol{\theta}_x$ of $k^x$ and the matrix $K^f$ to maximize the marginal likelihood $p(\mathbf{y}_o | X, \boldsymbol{\theta}_x, K^f)$. One way to achieve this is to use the fact that $\mathbf{y} | X \sim \mathcal{N}(\mathbf{0}, \Sigma)$. Therefore, gradient-based methods can be readily applied to maximize the marginal likelihood. In order to guarantee positive-semidefiniteness of $K^f$, one possible

parametrization is to use the Cholesky decomposition $K^f = LL^T$ where $L$ is lower triangular. Computing the derivatives of the marginal likelihood with respect to $L$ and $\boldsymbol{\theta}_x$ is straightforward. A drawback of this approach is its computational cost as it requires the inversion of a matrix of potential size $MN \times MN$ (or solving an $MN \times MN$ linear system) at each optimization step. Note, however, that one only needs to actually compute the Gram matrix and its inverse at the visible locations corresponding to $\mathbf{y}_o$.

Alternatively, it is possible to exploit the Kronecker product structure of the full covariance matrix as in [6], where an EM algorithm is proposed such that learning of $\boldsymbol{\theta}_x$ and $K^f$ in the M-step is decoupled. This has the advantage that closed-form updates for $K^f$ and $D$ can be obtained (see equation (5)), and that $K^f$ is guaranteed to be positive-semidefinite. The details of the EM algorithm are as follows: Let $\mathbf{f}$ be the vector of function values corresponding to $\mathbf{y}$, and similarly for $F$ wrt $Y$. Further, let $\mathbf{y}_{\cdot l}$ denote the vector $(y_{1l}, \ldots, y_{Nl})^T$ and similarly for $\mathbf{f}_{\cdot l}$. Given the missing data, which in this case is $\mathbf{f}$, the complete-data log-likelihood is

$$L_{\text{comp}} = -\frac{N}{2} \log |K^f| - \frac{M}{2} \log |K^x| - \frac{1}{2} \operatorname{tr}\left[\left(K^f\right)^{-1} F^T \left(K^x\right)^{-1} F\right]$$
$$- \frac{N}{2} \sum_{l=1}^{M} \log \sigma_l^2 - \frac{1}{2} \operatorname{tr}\left[(Y-F)D^{-1}(Y-F)^T\right] - \frac{MN}{2} \log 2\pi \quad (3)$$

from which we have following updates:

$$\widehat{\boldsymbol{\theta}}_x = \arg\min_{\boldsymbol{\theta}_x} \left(N \log \left|\left\langle F^T \left(K^x(\boldsymbol{\theta}_x)\right)^{-1} F\right\rangle\right| + M \log |K^x(\boldsymbol{\theta}_x)|\right) \quad (4)$$

$$\widehat{K}^f = N^{-1} \left\langle F^T \left(K^x(\widehat{\boldsymbol{\theta}}_x)\right)^{-1} F\right\rangle \qquad \widehat{\sigma}_l^2 = N^{-1} \left\langle (\mathbf{y}_{\cdot l} - \mathbf{f}_{\cdot l})^T (\mathbf{y}_{\cdot l} - \mathbf{f}_{\cdot l})\right\rangle \quad (5)$$

where the expectations $\langle \cdot \rangle$ are taken with respect to $p\left(\mathbf{f}|\mathbf{y}_o, \boldsymbol{\theta}_x, K^f\right)$, and $\widehat{\cdot}$ denotes the updated parameters. For clarity, let us consider the case where $\mathbf{y}_o = \mathbf{y}$, i.e. a block design. Then $p\left(\mathbf{f}|\mathbf{y}, \boldsymbol{\theta}_x, K^f\right) = \mathcal{N}\left((K^f \otimes K^x)\Sigma^{-1}\mathbf{y}, (K^f \otimes K^x) - (K^f \otimes K^x)\Sigma^{-1}(K^f \otimes K^x)\right).$

We have seen that $\Sigma$ needs to be inverted (in time $O(M^3N^3)$) for both making predictions and learning the hyperparameters (when considering noisy observations). This can lead to computational problems if $MN$ is large. In section 3 we give some approximations that can help speed up these computations.

## 2.3 Noiseless observations and the cancellation of inter-task transfer

One particularly interesting case to consider is noise-free observations at the same locations for all tasks (i.e. a block-design) so that $\mathbf{y}|X \sim \text{Normal}(\mathbf{0}, K^f \otimes K^x)$. In this case maximizing the marginal likelihood $p(\mathbf{y}|X)$ wrt the parameters $\boldsymbol{\theta}_x$ of $k^x$ reduces to maximizing $-M \log |K^x| - N \log |Y^T (K^x)^{-1} Y|$, an expression that does not depend on $K^f$. After convergence we can obtain $K^f$ as $\hat{K}^f = \frac{1}{N} Y^T (K^x)^{-1} Y$. The intuition behind is this: The responses $Y$ are correlated via $K^f$ and $K^x$. We can learn $K^f$ by decorrelating $Y$ with $(K^x)^{-1}$ first so that only correlation with respect to $K^f$ is left. Then $K^f$ is simply the sample covariance of the de-correlated $Y$.

Unfortunately, in this case there is effectively no transfer between the tasks (given the kernels). To see this, consider making predictions at a new location $\mathbf{x}_*$ for all tasks. We have (using the mixed-product property of Kronecker products) that

$$\overline{\mathbf{f}}(\mathbf{x}_*) = \left(K^f \otimes \mathbf{k}_*^x\right)^T \left(K^f \otimes K^x\right)^{-1} \mathbf{y} \quad (6)$$
$$= \left((K^f)^T \otimes (\mathbf{k}_*^x)^T\right) \left((K^f)^{-1} \otimes (K^x)^{-1}\right) \mathbf{y} \quad (7)$$
$$= \left[\left(K^f (K^f)^{-1}\right) \otimes \left((\mathbf{k}_*^x)^T (K^x)^{-1}\right)\right] \mathbf{y} \quad (8)$$
$$= \begin{pmatrix} (\mathbf{k}_*^x)^T (K^x)^{-1} \mathbf{y}_{\cdot 1} \\ \vdots \\ (\mathbf{k}_*^x)^T (K^x)^{-1} \mathbf{y}_{\cdot M} \end{pmatrix}, \quad (9)$$

and similarly for the covariances. Thus, in the noiseless case with a block design, the predictions for task $l$ depend only on the targets $\mathbf{y}_{\cdot l}$. In other words, there is a cancellation of transfer. One can

in fact generalize this result to show that the cancellation of transfer for task $l$ does still hold even if the observations are only sparsely observed at locations $X = (\mathbf{x}_1, \dots, \mathbf{x}_N)$ on the other tasks. After having derived this result we learned that it is known as *autokrigeability* in the geostatistics literature [7], and is also related to the *symmetric Markov property* of covariance functions that is discussed in [8]. We emphasize that if the observations are noisy, or if there is not a block design, then this result on cancellation of transfer will not hold. This result can also be generalized to multidimensional tensor product covariance functions and grids [9].

## 3 Approximations to speed up computations

The issue of dealing with large $N$ has been much studied in the GP literature, see [10, ch. 8] and [11] for overviews. In particular, one can use sparse approximations where only $Q$ out of $N$ data points are selected as *inducing inputs*[11]. Here, we use the Nyström approximation of $K^x$ in the marginal likelihood, so that $K^x \approx \widetilde{K}^x \overset{\text{def}}{=} K^x_{\cdot\mathcal{I}}(K^x_{\mathcal{I}\mathcal{I}})^{-1}K^x_{\mathcal{I}\cdot}$, where $\mathcal{I}$ indexes $Q$ rows/columns of $K^x$. In fact for the posterior at the training points this result is obtained from both the subset of regressors (SoR) and projected process (PP) approximations described in [10, ch. 8].

Specifying a full rank $K^f$ requires $M(M+1)/2$ parameters, and for large $M$ this would be a lot of parameters to estimate. One parametrization of $K^f$ that reduces this problem is to use a `PPCA` model [12] $K^f \approx \widetilde{K}^f \overset{\text{def}}{=} U\Lambda U^{\mathrm{T}} + s^2 I_M$, where $U$ is an $M \times P$ matrix of the $P$ principal eigenvectors of $K^f$, $\Lambda$ is a $P \times P$ diagonal matrix of the corresponding eigenvalues, and $s^2$ can be determined analytically from the eigenvalues of $K^f$ (see [12] and references therein). For numerical stability, we may further use the incomplete-Cholesky decomposition setting $U\Lambda U^{\mathrm{T}} = \tilde{L}\tilde{L}^{\mathrm{T}}$, where $\tilde{L}$ is a $M \times P$ matrix. Below we consider the case $s = 0$, i.e. a rank-$P$ approximation to $K^f$.

Applying both approximations to get $\Sigma \approx \widetilde{\Sigma} \overset{\text{def}}{=} \tilde{K}^f \otimes \tilde{K}^x + D \otimes I_N$, we have, after using the *Woodbury* identity, $\widetilde{\Sigma}^{-1} = \Delta^{-1} - \Delta^{-1}B\left[I \otimes K^x_{\mathcal{I}\mathcal{I}} + B^{\mathrm{T}}\Delta^{-1}B\right]^{-1}B^{\mathrm{T}}\Delta^{-1}$ where $B \overset{\text{def}}{=} (\tilde{L} \otimes K^x_{\cdot\mathcal{I}})$, and $\Delta \overset{\text{def}}{=} D \otimes I_N$ is a diagonal matrix. As $\tilde{K}^f \otimes \tilde{K}^x$ has rank $PQ$, we have that computation of $\widetilde{\Sigma}^{-1}\mathbf{y}$ takes $O(MNP^2Q^2)$.

For the `EM` algorithm, the approximation of $\widetilde{K}^x$ poses a problem in (4) because for the rank-deficient matrix $\widetilde{K}^x$, its log-determinant is negative infinity, and its matrix inverse is undefined. We overcome this by considering $\widetilde{K}^x = \lim_{\xi \to 0}(K^x_{\cdot\mathcal{I}}(K^x_{\mathcal{I}\mathcal{I}})^{-1}K^x_{\mathcal{I}\cdot} + \xi^2 I)$, so that we solve an equivalent optimization problem where the log-determinant is replaced by the well-defined $\log|K^x_{\mathcal{I}\cdot}K^x_{\cdot\mathcal{I}}| - \log|K^x_{\mathcal{I}\mathcal{I}}|$, and the matrix inverse is replaced by the pseudo-inverse. With these approximations the computational complexity of hyperparameter learning can be reduced to $O(MNP^2Q^2)$ per iteration for both the Cholesky and EM methods.

## 4 Related work

There has been a lot of work in recent years on multi-task learning (or inductive transfer) using methods such as Neural Networks, Gaussian Processes, Dirichlet Processes and Support Vector Machines, see e.g. [2, 13] for early references. The key issue concerns what properties or aspects should be shared across tasks. Within the GP literature, [14, 15, 16, 17, 18] give models where the covariance matrix of the full (noiseless) system is block diagonal, and each of the $M$ blocks is induced from the same kernel function. Under these models each $\mathbf{y}_{\cdot i}$ is conditionally independent, but inter-task tying takes place by sharing the kernel function across tasks. In contrast, in our model and in [5, 3, 4] the covariance is not block diagonal.

The semiparametric latent factor model (SLFM) of Teh et al [5] involves having $P$ latent processes (where $P \leq M$) and each of these latent processes has its own covariance function. The noiseless outputs are obtained by linear mixing of these processes with a $M \times P$ matrix $\Phi$. The covariance matrix of the system under this model has rank at most $PN$, so that when $P < M$ the system corresponds to a degenerate GP. Our model is similar to [5] but simpler, in that all of the $P$ latent processes share the same covariance function; this reduces the number of free parameters to be fitted and should help to minimize overfitting. With a common covariance function $k^x$, it turns out that $K^f$ is equal to $\Phi\Phi^{\mathrm{T}}$, so a $K^f$ that is strictly positive definite corresponds to using $P = M$ latent

processes. Note that if $P > M$ one can always find an $M \times M$ matrix $\Phi'$ such that $\Phi'\Phi'^{\mathrm{T}} = \Phi\Phi^{\mathrm{T}}$. We note also that the approximation methods used in [5] are different to ours, and were based on the subset of data (SoD) method using the informative vector machine (IVM) selection heuristic.

In the geostatistics literature, the prior model for $f$. given in eq. (1) is known as the *intrinsic correlation model* [7], a specific case of *co-kriging*. A sum of such processes is known as the *linear coregionalization model* (LCM) [7] for which [6] gives an EM-based algorithm for parameter estimation. Our model for the observations corresponds to an LCM model with two processes: the process for $f$. and the noise process. Note that SLFM can also be seen as an instance of the LCM model. To see this, let $E_{pp}$ be a $P \times P$ diagonal matrix with 1 at $(p,p)$ and zero elsewhere. Then we can write the covariance in SLFM as $(\Phi \otimes I)(\sum_{p=1}^{P} E_{pp} \otimes K_p^x)(\Phi \otimes I)^{\mathrm{T}} = \sum_{p=1}^{P}(\Phi E_{pp}\Phi^{\mathrm{T}}) \otimes K_p^x$, where $\Phi E_{pp}\Phi^{\mathrm{T}}$ is of rank 1.

Evgeniou et al. [19] consider methods for inducing correlations between tasks based on a correlated prior over linear regression parameters. In fact this corresponds to a GP prior using the kernel $k(\mathbf{x}, \mathbf{x}') = \mathbf{x}^T A \mathbf{x}'$ for some positive definite matrix $A$. In their experiments they use a restricted form of $K^f$ with $K_{lk}^f = (1 - \lambda) + \lambda M \delta_{lk}$ (their eq. 25), i.e. a convex combination of a rank-1 matrix of ones and a multiple of the identity. Notice the similarity to the PPCA form of $K^f$ given in section 3.

## 5    Experiments

We evaluate our model on two different applications. The first application is a compiler performance prediction problem where the goal is to predict the *speed-up* obtained in a given program (task) when applying a sequence of code transformations $\mathbf{x}$. The second application is an exam score prediction problem where the goal is to predict the *exam score* obtained by a student $\mathbf{x}$ belonging to a specific school (task). In the sequel, we will refer to the data related to the first problem as the *compiler data* and the data related to the second problem as the *school data*.

We are interested in assessing the benefits of our approach not only with respect to the no-transfer case but also with respect to the case when a parametric GP is used on the joint input-dependent and task-dependent space as in [3]. To train the parametric model note that the parameters of the covariance function over task descriptors $k^f(\mathbf{t}, \mathbf{t}')$ can be tuned by maximizing the marginal likelihood, as in [3]. For the free-form $K^f$ we initialize this (given $k^x(\cdot, \cdot)$) by using the noise-free expression $\hat{K}^f = \frac{1}{N} Y^T (K^x)^{-1} Y$ given in section 2.3 (or the appropriate generalization when the design is not complete). For both applications we have used a squared-exponential (or Gaussian) covariance function $k^x$ and a non-parametric form for $K^f$. Where relevant the parametric covariance function $k^f$ was also taken to be of squared-exponential form. Both $k^x$ and $k^f$ used an automatic relevance determination (ARD) parameterization, i.e. having a length scale for each feature dimension. All the length scales in $k^x$ and $k^f$ were initialized to 1, and all $\sigma_l^2$ were constrained to be equal for all tasks and initialized to 0.01.

### 5.1    Description of the Data

**Compiler Data**. This data set consists of 11 C programs for which an exhaustive set of 88214 sequences of code transformations have been applied and their corresponding speed-ups have been recorded. Each task is to predict the speed-up on a given program when applying a specific transformation sequence. The speed-up after applying a transformation sequence on a given program is defined as the ratio of the execution time of the original program (baseline) over the execution time of the transformed program. Each transformation sequence is described as a 13-dimensional vector $\mathbf{x}$ that records the absence/presence of one-out-of 13 single transformations. In [3] the task-descriptor features (for each program) are based on the speed-ups obtained on a pre-selected set of 8 transformations sequences, so-called "canonical responses". The reader is referred to [3, section 3] for a more detailed description of the data.

**School Data**. This data set comes from the Inner London Education Authority (ILEA) and has been used to study the effectiveness of schools. It is publicly available under the name of "school effectiveness" at `http://www.cmm.bristol.ac.uk/learning-training/multilevel-m-support/datasets.shtml`. It consists of examination records from 139

secondary schools in years 1985, 1986 and 1987. It is a random $50\%$ sample with 15362 students. This data has also been used in the context of multi-task learning by Bakker and Heskes [20] and Evgeniou et al. [19]. In [20] each task is defined as the prediction of the exam score of a student belonging to a specific school based on four student-dependent features (year of the exam, gender, VR band and ethnic group) and four school-dependent features (percentage of students eligible for free school meals, percentage of students in VR band 1, school gender and school denomination). For comparison with [20, 19] we evaluate our model following the set up described above and similarly, we have created dummy variables for those features that are categorical forming a total of 19 student-dependent features and 8 school-dependent features. However, we note that school-descriptor features such as the percentage of students eligible for free school meals and the percentage of students in VR band 1 actually depend on the year the particular sample was taken.

It is important to emphasize that for both data sets there are task-descriptor features available. However, as we have described throughout this paper, our approach learns task similarity directly without the need for task-dependent features. Hence, we have neglected these features in the application of our free-form $K^f$ method.

# 6   Results

For the compiler data we have $M = 11$ tasks and we have used a Cholesky decomposition $K^f = LL^T$. For the school data we have $M = 139$ tasks and we have preferred a reduced rank parameterization of $K^f \approx \widetilde{K}^f = \widetilde{L}\widetilde{L}^T$, with ranks 1, 2, 3 and 5. We have learnt the parameters of the models so as to maximize the marginal likelihood $p(\mathbf{y}_o|X, K^f, \boldsymbol{\theta}_x)$ using gradient-based search in MATLAB with Carl Rasmussen's minimize.m. In our experiments this method usually outperformed EM in the quality of solutions found and in the speed of convergence.

**Compiler Data:** For this particular application, in a real-life scenario it is critical to achieve good performance with a low number of training data-points per task given that a training data-point requires the compilation and execution of a (potentially) different version of a program. Therefore, although there are a total of 88214 training points per program we have followed a similar set up to [3] by considering $N = 16, 32, 64$ and 128 transformation sequences per program for training. All the $M = 11$ programs (tasks) have been used for training, and predictions have been done at the (unobserved) remaining $88214 - N$ inputs. For comparison with [3] the mean absolute error (between the actual speed-ups of a program and the predictions) has been used as the measure of performance. Due to the variability of the results depending on training set selection we have considered 10 different replications.

Figure 1 shows the mean absolute errors obtained on the compiler data for some of the tasks (top row and bottom left) and on average for all the tasks (bottom right). *Sample task* 1 (histogram) is an example where learning the tasks simultaneously brings major benefits over the no transfer case. Here, multi-task GP (transfer free-form) provides a reduction on the mean absolute error of up to 6 times. Additionally, it is consistently (although only marginally) superior to the parametric approach. For *sample task* 2 (fir), our approach not only significantly outperforms the no transfer case but also provides greater benefits over the parametric method (which for $N = 64$ and 128 is worse than no transfer). *Sample task* 3 (adpcm) is the only case out of all 11 tasks where our approach degrades performance, although it should be noted that all the methods perform similarly. Further analysis of the data indicates that learning on this task is hard as there is a lot of variability that cannot be explained by the 1-out-of-13 encoding used for the input features. Finally, for *all tasks* on average (bottom right) our approach brings significant improvements over single task learning and consistently outperforms the parametric method. For all tasks except one our model provides better or roughly equal performance than the non-transfer case and the parametric model.

**School Data:** For comparison with [20, 19] we have made 10 random splits of the data into training ($75\%$) data and test ($25\%$) data. Due to the categorical nature of the data there are a maximum of $N = 202$ different student-dependent feature vectors $\mathbf{x}$. Given that there can be multiple observations of a target value for a given task at a specific input $\mathbf{x}$, we have taken the mean of these observations and corrected the noise variances by dividing them over the corresponding number of observations. As in [19], the percentage explained variance is used as the measure of performance. This measure can be seen as the percentage version of the well known coefficient of determination $r^2$ between the actual target values and the predictions.

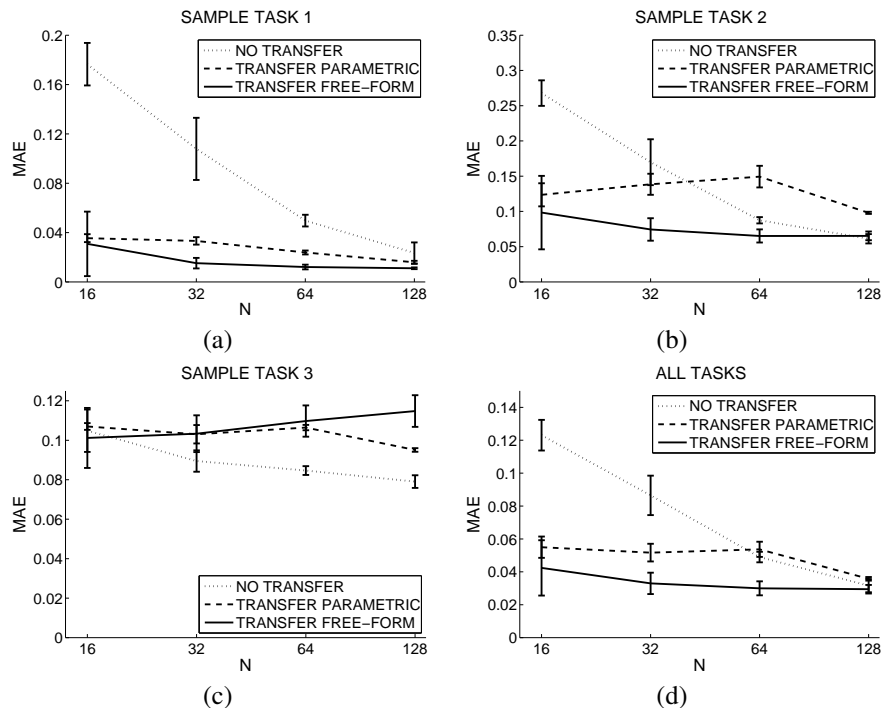

Figure 1: Panels (a), (b) and (c) show the average mean absolute error on the compiler data as a function of the number of training points for specific tasks. *no transfer* stands for the use of a single GP for each task separately; *transfer parametric* is the use of a GP with a joint parametric (SE) covariance function as in [3]; and *transfer free-form* is multi-task GP with a "free form" covariance matrix over tasks. The error bars show $\pm$ one standard deviation taken over the 10 replications. Panel (d) shows the average MAE over all 11 tasks, and the error bars show the average of the standard deviations over all 11 tasks.

The results are shown in Table 1; note that larger figures are better. The parametric result given in the table was obtained from the school-descriptor features; in the cases where these features varied for a given school over the years, an average was taken. The results show that better results can be obtained by using multi-task learning than without. For the non-parametric $K^f$, we see that the rank-2 model gives best performance. This performance is also comparable with the best (29.5%) found in [20]. We also note that our *no transfer* result of 21.1% is much better than the baseline of 9.7% found in [20] using neural networks.

| no transfer | parametric | rank 1 | rank 2 | rank 3 | rank 5 |
|---|---|---|---|---|---|
| 21.05 (1.15) | 31.57 (1.61) | 27.02 (2.03) | 29.20 (1.60) | 24.88 (1.62) | 21.00 (2.42) |

Table 1: Percentage variance explained on the school dataset for various situations. The figures in brackets are standard deviations obtained from the ten replications.

On the school data the parametric approach for $K^f$ slightly outperforms the non-parametric method, probably due to the large size of this matrix relative to the amount of data. One can also run the parametric approach creating a task for every unique school-features descriptor[1]; this gives rise to 288 tasks rather than 139 schools, and a performance of 33.08% ($\pm 1.57$). Evgeniou et al [19] use a linear predictor on all 8 features (i.e. they combine both student and school features into $\mathbf{x}$) and then introduce inter-task correlations as described in section 4. This approach uses the same information as our 288 task case, and gives similar performance of around 34% (as shown in Figure 3 of [19]).

# 7 Conclusion

In this paper we have described a method for multi-task learning based on a GP prior which has inter-task correlations specified by the task similarity matrix $K^f$. We have shown that in a noise-free block design, there is actually a cancellation of transfer in this model, but not in general. We have successfully applied the method to the compiler and school problems. An advantage of our method is that task-descriptor features are not required (c.f. [3, 4]). However, such features might be beneficial if we consider a setup where there are only few datapoints for a new task, and where the task-descriptor features convey useful information about the tasks.

## Acknowledgments

CW thanks Dan Cornford for pointing out the prior work on autokrigeability. KMC thanks DSO NL for support. This work is supported under EPSRC grant GR/S71118/01 , EU FP6 STREP MILEPOST IST-035307, and in part by the IST Programme of the European Community, under the PASCAL Network of Excellence, IST-2002-506778. This publication only reflects the authors' views.

## Footnotes

[1]Recall from section 5.1 that the school features can vary over different years.

## References

[1] Jonathan Baxter. A Model of Inductive Bias Learning. *JAIR*, 12:149–198, March 2000.

[2] Rich Caruana. Multitask Learning. *Machine Learning*, 28(1):41–75, July 1997.

[3] Edwin V. Bonilla, Felix V. Agakov, and Christopher K. I. Williams. Kernel Multi-task Learning using Task-specific Features. In *Proceedings of the 11th AISTATS*, March 2007.

[4] Kai Yu, Wei Chu, Shipeng Yu, Volker Tresp, and Zhao Xu. Stochastic Relational Models for Discriminative Link Prediction. In *NIPS 19*, Cambridge, MA, 2007. MIT Press.

[5] Yee Whye Teh, Matthias Seeger, and Michael I. Jordan. Semiparametric latent factor models. In *Proceedings of the 10th AISTATS*, pages 333–340, January 2005.

[6] Hao Zhang. Maximum-likelihood estimation for multivariate spatial linear coregionalization models. *Environmetrics*, 18(2):125–139, 2007.

[7] Hans Wackernagel. *Multivariate Geostatistics: An Introduction with Applications*. Springer-Verlag, Berlin, 2nd edition, 1998.

[8] A. O'Hagan. A Markov property for covariance structures. Statistics Research Report 98-13, Nottingham University, 1998.

[9] C. K. I. Williams, K. M. A. Chai, and E. V. Bonilla. A note on noise-free Gaussian process prediction with separable covariance functions and grid designs. Technical report, University of Edinburgh, 2007.

[10] C. E. Rasmussen and C. K. I. Williams. *Gaussian Processes for Machine Learning*. MIT Press, Cambridge, Massachusetts, 2006.

[11] Joaquin Quiñonero-Candela, Carl Edward Rasmussen, and Christopher K. I. Williams. Approximation Methods for Gaussian Process Regression. In *Large Scale Kernel Machines*. MIT Press, 2007. To appear.

[12] Michael E. Tipping and Christopher M. Bishop. Probabilistic principal component analysis. *Journal of the Royal Statistical Society, Series B*, 61(3):611–622, 1999.

[13] S. Thrun. Is Learning the $n$-th Thing Any Easier Than Learning the First? In *NIPS 8*, 1996.

[14] Thomas P. Minka and Rosalind W. Picard. Learning How to Learn is Learning with Point Sets. 1999.

[15] Neil D. Lawrence and John C. Platt. Learning to learn with the Informative Vector Machine. In *Proceedings of the 21st International Conference on Machine Learning*, July 2004.

[16] Kai Yu, Volker Tresp, and Anton Schwaighofer. Learning Gaussian Processes from Multiple Tasks. In *Proceedings of the 22nd International Conference on Machine Learning*, 2005.

[17] Anton Schwaighofer, Volker Tresp, and Kai Yu. Learning Gaussian Process Kernels via Hierarchical Bayes. In *NIPS 17*, Cambridge, MA, 2005. MIT Press.

[18] Shipeng Yu, Kai Yu, Volker Tresp, and Hans-Peter Kriegel. Collaborative Ordinal Regression. In *Proceedings of the 23rd International Conference on Machine Learning*, June 2006.

[19] Theodoros Evgeniou, Charles A. Micchelli, and Massimiliano Pontil. Learning Multiple Tasks with Kernel Methods. *Journal of Machine Learning Research*, 6:615–537, April 2005.

[20] Bart Bakker and Tom Heskes. Task Clustering and Gating for Bayesian Multitask Learning. *Journal of Machine Learning Research*, 4:83–99, May 2003.
